# An Asynchronous Hidden Markov Model for Audio-Visual Speech Recognition

**Samy Bengio**
Dalle Molle Institute for Perceptual Artificial Intelligence (IDIAP)
CP 592, rue du Simplon 4,
1920 Martigny, Switzerland
bengio@idiap.ch, http://www.idiap.ch/~bengio

## Abstract

This paper presents a novel Hidden Markov Model architecture to model the joint probability of pairs of asynchronous sequences describing the same event. It is based on two other Markovian models, namely Asynchronous Input/Output Hidden Markov Models and Pair Hidden Markov Models. An EM algorithm to train the model is presented, as well as a Viterbi decoder that can be used to obtain the optimal state sequence as well as the alignment between the two sequences. The model has been tested on an audio-visual speech recognition task using the M2VTS database and yielded robust performances under various noise conditions.

## 1 Introduction

Hidden Markov Models (HMMs) are statistical tools that have been used successfully in the last 30 years to model difficult tasks such as speech recognition [6] or biological sequence analysis [4]. They are very well suited to handle discrete of continuous sequences of varying sizes. Moreover, an efficient training algorithm (EM) is available, as well as an efficient decoding algorithm (Viterbi), which provides the optimal sequence of states (and the corresponding sequence of high level events) associated with a given sequence of low-level data.

On the other hand, multimodal information processing is currently a very challenging framework of applications including multimodal person authentication, multimodal speech recognition, multimodal event analyzers, etc. In that framework, the same sequence of events is represented not only by a single sequence of data but by a series of sequences of data, each of them coming eventually from a different *modality*: video streams with various viewpoints, audio stream(s), etc.

One such task, which will be presented in this paper, is multimodal speech recognition using both a microphone and a camera recording a speaker simultaneously while he (she) speaks. It is indeed well known that seeing the speaker's face in addition to hearing his (her) voice can often improve speech intelligibility, particularly in noisy environments [7], mainly thanks to the complementarity of the visual and acoustic signals. Previous solutions proposed for this task can be subdivided into

two categories [8]: *early integration*, where both signals are first modified to reach the same frame rate and are then modeled jointly, or *late integration*, where the signals are modeled separately and are combined later, during decoding. While in the former solution, the alignment between the two sequences is decided *a priori*, in the latter, there is no explicit learning of the joint probability of the two sequences. An example of late integration is presented in [3], where the authors present a multi-stream approach where each stream is modeled by a different HMM, while decoding is done on a combined HMM (with various combination approaches proposed).

In this paper, we present a novel Asynchronous Hidden Markov Model (AHMM) that can learn the joint probability of pairs of sequences of data representing the same sequence of events, even when the events are not synchronized between the sequences. In fact, the model enables to *desynchronize* the streams by temporarily stretching one of them in order to obtain a better match between the corresponding frames. The model can thus be directly applied to the problem of audio-visual speech recognition where sometimes lips start to move before any sound is heard for instance. The paper is organized as follows: in the next section, the AHMM model is presented, followed by the corresponding EM training and Viterbi decoding algorithms. Related models are then presented and implementation issues are discussed. Finally, experiments on a audio-visual speech recognition task based on the M2VTS database are presented, followed by a conclusion.

## 2    The Asynchronous Hidden Markov Model

For the sake of simplicity, let us present here the case where one is interested in modeling the joint probability of 2 asynchronous sequences, denoted $x_1^T$ and $y_1^S$ with $S \leq T$ without loss of generality[1].

We are thus interested in modeling $p(x_1^T, y_1^S)$. As it is intractable if we do it directly by considering all possible combinations, we introduce a hidden variable $q$ which represents the *state* as in the classical HMM formulation, and which is synchronized with the longest sequence. Let $N$ be the number of states.

Moreover, in the model presented here, we always emit $x_t$ at time $t$ and sometimes emit $y_s$ at time $t$. Let us first define $\epsilon(i, t) = P(\tau_t{=}s|\tau_{t-1}{=}s-1, q_t{=}i, x_1^t, y_1^s)$ as the probability that the system emits the next observation of sequence $y$ at time $t$ while in state $i$. The additional hidden variable $\tau_t = s$ can be seen as the alignment between $y$ and $q$ (and $x$ which is aligned with $q$). Hence, we model $p(x_1^T, y_1^S, q_1^T, \tau_1^T)$.

### 2.1    Likelihood Computation

Using classical HMM independence assumptions, a simple **forward procedure** can be used to compute the joint likelihood of the two sequences, by introducing the following $\alpha$ intermediate variable for each state and each possible alignment between the sequences $x$ and $y$:

$$\alpha(i, s, t) = p(q_t{=}i, \tau_t{=}s, x_1^t, y_1^s) \tag{1}$$

$$\alpha(i, s, t) = \epsilon(i, t)p(x_t, y_s|q_t{=}i) \sum_{j=1}^{N} P(q_t{=}i|q_{t-1}{=}j)\alpha(j, s-1, t-1)$$

$$+ \quad (1 - \epsilon(i,t))p(x_t|q_t{=}i) \sum_{j=1}^{N} P(q_t{=}i|q_{t-1}{=}j)\alpha(j,s,t-1)$$

which is very similar to the corresponding $\alpha$ variable used in normal HMMs[2]. It can then be used to compute the joint likelihood of the two sequences as follows:

$$
\begin{aligned}
p(x_1^T, y_1^S) &= \sum_{i=1}^{N} p(q_T{=}i, \tau_T{=}S, x_1^T, y_1^S) \qquad (2) \\
&= \sum_{i=1}^{N} \alpha(i, S, T) \,.
\end{aligned}
$$

## 2.2 Viterbi Decoding

Using the same technique and replacing all the sums by max operators, a Viterbi decoding algorithm can be derived in order to obtain the most probable path along the sequence of states and alignments between $x$ and $y$:

$$
\begin{aligned}
V(i,s,t) &= \max_{\tau_1^{t-1}, q_1^{t-1}} p(q_t{=}i, \tau_t{=}s, x_1^t, y_1^s) \qquad (3) \\
&= \max \begin{cases} (\epsilon(i,t)p(x_t, y_s|q_t{=}i) \max_{j} P(q_t{=}i|q_{t-1}{=}j)V(j,s-1,t-1), \\ (1-\epsilon(i,t))p(x_t|q_t{=}i) \max_{j} P(q_t{=}i|q_{t-1}{=}j)V(j,s,t-1)) \end{cases} .
\end{aligned}
$$

The best path is then obtained after having computed $V(i, S, T)$ for the best final state $i$ and backtracking along the best path that could reach it[3].

## 2.3 An EM Training Algorithm

An EM training algorithm can also be derived in the same fashion as in classical HMMs. We here sketch the resulting algorithm, without going into more details[4].

**Backward Step:** Similarly to the forward step based on the $\alpha$ variable used to compute the joint likelihood, a backward variable, $\beta$ can also be derived as follows:

$$
\beta(i,s,t) = p(x_{t+1}^T, y_{s+1}^S|q_t{=}i, \tau_t{=}s) \qquad (4)
$$

$$
\beta(i,s,t) = \sum_{j=1}^{N} \epsilon(j, t+1)p(x_{t+1}, y_{s+1}|q_{t+1}{=}j)P(q_{t+1}{=}j|q_t{=}i)\beta(j, s+1, t+1)
$$

$$
+ \sum_{j=1}^{N} (1 - \epsilon(j, t+1))p(x_{t+1}|q_{t+1}{=}j)P(q_{t+1}{=}j|q_t{=}i)\beta(j, s, t+1) \,.
$$

**E-Step:** Using both the forward and backward variables, one can compute the posterior probabilities of the hidden variables of the system, namely the posterior on the state when it emits on both sequences, the posterior on the state when it emits on $x$ only, and the posterior on transitions.

Let $\alpha^1(i, s, t)$ be the part of $\alpha(i, s, t)$ when state $i$ emits on $y$ at time $t$:

$$\alpha^1(i, s, t) \;=\; \epsilon(i,t)p(x_t, y_s|q_t{=}i)\sum_{j=1}^{N} P(q_t{=}i|q_{t-1}{=}j)\alpha(j, s-1, t-1) \quad (5)$$

and similarly, let $\alpha^0(i, s, t)$ be the part of $\alpha(i, s, t)$ when state $i$ does not emit on $y$ at time $t$:

$$\alpha^0(i, s, t) \;=\; (1 - \epsilon(i,t))p(x_t|q_t{=}i)\sum_{j=1}^{N} P(q_t{=}i|q_{t-1}{=}j)\alpha(j, s, t-1) \;. \quad (6)$$

Then the posterior on state $i$ when it emits joint observations of sequences $x$ and $y$ is

$$P(q_t{=}i, \tau_t{=}s|\tau_{t-1}{=}s-1, x_1^T, y_1^S) = \frac{\alpha^1(i, s, t)\beta(i, s, t)}{P(x_1^T, y_1^S)} \;, \quad (7)$$

the posterior on state $i$ when it emits the next observation of sequence $x$ only is

$$P(q_t{=}i, \tau_t{=}s|\tau_{t-1}{=}s, x_1^T, y_1^S) = \frac{\alpha^0(i, s, t)\beta(i, s, t)}{P(x_1^T, y_1^S)} \;, \quad (8)$$

and the posterior on the transition between states $i$ and $j$ is

$$P(q_t{=}i, q_{t-1}{=}j|x_1^T, y_1^S) \;=\; \frac{P(q_t{=}i|q_{t-1}{=}j)}{P(x_1^T, y_1^S)} \cdot \left( \begin{array}{l} \displaystyle\sum_{s=1}^{S} \alpha(j, s-1, t-1)p(x_t, y_s|q_t{=}i)\epsilon(i,t)\beta(i, s, t) + \\[2ex] \displaystyle\sum_{s=0}^{S} \alpha(j, s, t-1)p(x_t|q_t{=}i)(1 - \epsilon(i,t))\beta(i, s, t) \end{array} \right) \;. \quad (9)$$

**M-Step:** The Maximization step is performed exactly as in normal HMMs: when the distributions are modeled by exponential functions such as Gaussian Mixture Models, then an exact maximization can be performed using the posteriors. Otherwise, a Generalized EM is performed by gradient ascent, back-propagating the posteriors through the parameters of the distributions.

## 3   Related Models

The present AHMM model is related to the *Pair HMM* model [4], which was proposed to search for the best alignment between two DNA sequences. It was thus designed and used mainly for discrete sequences. Moreover, the architecture of the Pair HMM model is such that a given state is designed to always emit either one OR two vectors, while in the proposed AHMM model, each state can always emit both one or two vectors, depending on $\epsilon(i,t)$, which is learned. In fact, when $\epsilon(i,t)$ is deterministic and solely depends on $i$, we can indeed recover the Pair HMM model by slightly transforming the architecture.

It is also very similar to the asynchronous version of *Input/Output HMMs* [2], which was proposed for speech recognition applications. The main difference here is that in

AHMMs both sequences are considered as output, while in Asynchronous IOHMMs one of the sequence (the shorter one, the output) is conditioned on the other one (the input). The resulting Viterbi decoding algorithm is thus different since in Asynchronous IOHMMs one of the sequence, the input, is known during decoding, which is not the case in AHMMs.

# 4 Implementation Issues

## 4.1 Time and Space Complexity

The proposed algorithms (either training or decoding) have a complexity of $\mathcal{O}(N^2 ST)$ where $N$ is the number of states (and assuming the worst case with ergodic connectivity), $S$ is the length of sequence $y$ and $T$ is the length of sequence $x$. This can become quickly intractable if both $x$ and $y$ are longer than, say, 1000 frames. It can however be shortened when *a priori* knowledge is known about possible alignments between $x$ and $y$. For instance, one can force the alignment between $x_t$ and $y_s$ to be such that $|t - \frac{T}{S}s| < k$ where $k$ is a constant representing the maximum stretching allowed between $x$ and $y$, which should not depend on $S$ nor $T$. In that case, the complexity (both in time and space) becomes $\mathcal{O}(N^2 T k)$, which is $k$ times the usual HMM training/decoding complexity.

## 4.2 Distributions to Model

In order to implement this system, we thus need to model the following distributions:

- $P(q_t{=}i|q_{t-1}{=}j)$: the transition distribution, as in normal HMMs;
- $p(x_t|q_t{=}i)$: the emission distribution in the case where only $x$ is emitted, as in normal HMMs;
- $p(x_t, y_s|q_t{=}i)$: the emission distribution in the case where both sequences are emitted. This distribution could be implemented in various forms, depending on the assumptions made on the data:
  - $x$ and $y$ are independent given state $i$:
  $$p(x_t, y_s|q_t{=}i) = p(x_t|q_t{=}i)p(y_s|q_t{=}i) \tag{10}$$
  - $y$ is conditioned on $x$:
  $$p(x_t, y_s|q_t{=}i) = p(y_s|x_t, q_t{=}i)p(x_t|q_t{=}i) \tag{11}$$
  - the joint probability is modeled directly, eventually forcing some common parameters from $p(x_t|q_t{=}i)$ and $p(x_t, y_s|q_t{=}i)$ to be shared.

  In the experiments described later in the paper, we have chosen the latter implementation, with no sharing except during initialization;
- $\epsilon(i, t) = P(\tau_t{=}s|\tau_{t-1}{=}s-1, q_t{=}i, x_1^t, y_1^s)$: the probability to emit on sequence $y$ at time $t$ on state $i$. With various assumptions, this probability could be represented as either independent on $i$, independent on $s$, independent on $x_t$ and $y_s$. In the experiments described later in the paper, we have chosen the latter implementation.

# 5 Experiments

Audio-visual speech recognition experiments were performed using the M2VTS database [5], which contains 185 recordings of 37 subjects, each containing acoustic

and video signals of the subject pronouncing the French digits from zero to nine. The video consisted of 286x360 pixel color images with a 25 Hz frame rate, while the audio was recorded at 48 kHz using a 16 bit PCM coding. Although the M2VTS database is one of the largest databases of its type, it is still relatively small compared to reference audio databases used in speech recognition. Hence, in order to increase the significance level of the experimental results, a 5-fold cross-validation method was used. Note that all the subjects always pronounced the same sequence of words but this information was not used during recognition[5].

The audio data was down-sampled to 8khz and every 10ms a vector of 16 MFCC coefficients and their first derivative, as well as the derivative of the log energy was computed, for a total of 33 features. Each image of the video stream was coded using 12 shape features and 12 intensity features, as described in [3]. The first derivative of each of these features was also computed, for a total of 48 features.

The HMM topology was as follows: we used left-to-right HMMs for each instance of the vocabulary, which consisted of the following 11 words: zero, un, deux trois, quatre, cinq, six, sept, huit, neuf, silence. Each model had between 3 to 9 states including non-emitting begin and end states.

In each emitting state, there was 3 distributions: $P(x_t|q_t)$, the emission distribution of audio-only data, which consisted of a Gaussian mixture of 10 Gaussians (of dimension 33), $P(x_t, y_s|q_t)$, the joint emission distribution of audio and video data, which consisted also of a Gaussian mixture of 10 Gaussians (of dimension 33+48=81), and $\epsilon(i, t)$, the probability that the system should emit on the video sequence, which was implemented for these preliminary experiments as a simple table.

Training was done using the EM algorithm described in the paper. However, in order to keep the computational time tractable, a constraint was imposed in the alignment between the audio and video streams: we did not consider alignments where audio and video information were farther than 0.5 second from each other.

Comparisons were made between the AHMM (taking into account audio and video), and a normal HMM taking into account either the audio or the video only. We also compared the model with a normal HMM trained on both audio and video streams manually synchronized (each frame of the video stream was repeated in multiple copies in order to reach the same rate as the audio stream). Moreover, in order to show the interest of robust multimodal speech recognition, we injected various levels of noise in the audio stream during decoding (training was always done using clean audio). The noise was taken from the Noisex database [9], and was injected in order to reach signal-to-noise ratios of 10dB, 5dB and 0dB.

Note that all the hyper-parameters of these systems, such as the number of Gaussians in the mixtures, the number of EM iterations, or the minimum value of the variances of the Gaussians, were not tuned using the M2VTS dataset. They were taken from a previously trained model on a different task, Numbers'95.

Figure 1 and Table 1 present the results. As it can be seen, the AHMM yielded better results as soon as the noise level was significant (for clean data, the performance using the audio stream only was almost perfect, hence no enhancement was expected). Moreover, it never deteriorated significantly (using a 95% confidence interval) under the level of the video stream, no matter the level of noise in the audio stream.

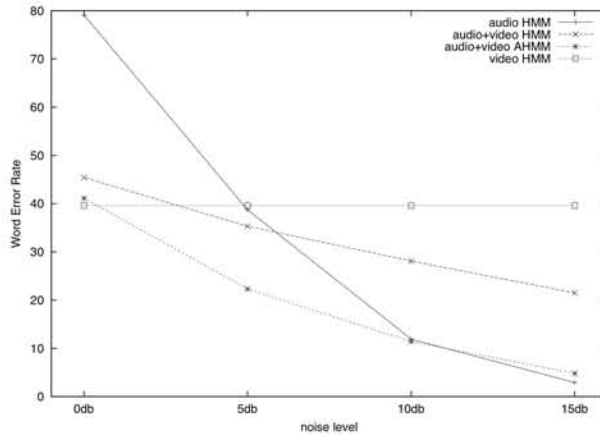

Figure 1: Word Error Rates (in percent, the lower the better), of various systems under various noise conditions during decoding (from 15 to 0 dB additive noise). The proposed model is the AHMM using both audio and video streams.

| Observations | Model | WER (%) and 95% CI | | | |
|---|---|---|---|---|---|
| | | 15 dB | 10 dB | 5 dB | 0 dB |
| audio | HMM | 2.9 ($\pm$ 2.4) | 11.9 ($\pm$ 4.7) | 38.7 ($\pm$ 7.1) | 79.1 ($\pm$ 5.9) |
| audio+video | HMM | 21.5 ($\pm$ 6.0) | 28.1 ($\pm$ 6.5) | 35.3 ($\pm$ 6.9) | 45.4 ($\pm$ 7.2) |
| audio+video | AHMM | 4.8 ($\pm$ 3.1) | 11.4 ($\pm$ 4.6) | 22.3 ($\pm$ 6.0) | 41.1 ($\pm$ 7.1) |

Table 1: Word Error Rates (WER, in percent, the lower the better) and corresponding Confidence Intervals (CI, in parenthesis), of various systems under various noise conditions during decoding (from 15 to 0 dB additive noise). The proposed model is the AHMM using both audio and video streams. An HMM using the clean video data only obtains 39.6% WER ($\pm$ 7.1).

An interesting side effect of the model is to provide an optimal alignment between the audio and the video streams. Figure 2 shows the alignment obtained while decoding sequence cd01 on data corrupted with 10dB Noisex noise. It shows that the rate between video and audio is far from being constant (it would have followed the stepped line) and hence computing the joint probability using the AHMM appears more informative than using a naive alignment and a normal HMM.

## 6   Conclusion

In this paper, we have presented a novel asynchronous HMM architecture to handle multiple sequences of data representing the same sequence of events. The model was inspired by two other well-known models, namely Pair HMMs and Asynchronous IOHMMs. An EM training algorithm was derived as well as a Viterbi decoding algorithm, and speech recognition experiments were performed on a multimodal database, yielding significant improvements on noisy audio data. Various propositions were made to implement the model but only the simplest ones were tested in this paper. Other solutions should thus be investigated soon. Moreover, other applications of the model should also be investigated, such as multimodal authentication.

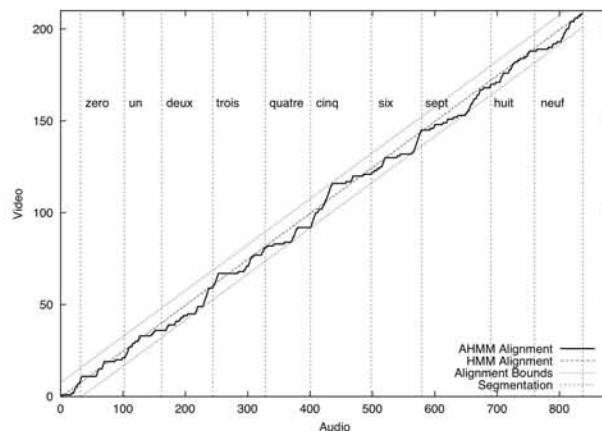

Figure 2: Alignment obtained by the model between video and audio streams on sequence `cd01` corrupted with a 10dB Noisex noise. The vertical lines show the obtained segmentation between the words. The stepped line represents a constant alignment.

## Acknowledgments

This research has been partially carried out in the framework of the European project LAVA, funded by the Swiss OFES project number 01.0412. The Swiss NCCR project IM2 has also partly funded this research. The author would like to thank Stephane Dupont for providing the extracted visual features and the experimental protocol used in the paper.

## Footnotes

[1]In fact, we assume that for all pairs of sequences $(x, y)$, the sequence $x$ is always at least as long as the sequence $y$. If this is not the case, a straightforward extension of the proposed model is then necessary.

[2]The full derivations are not given in this paper but can be found in the appendix of [1].

[3]In the case where one is only interested in the best state sequence (no matter the alignment), the solution is then to marginalize over all the alignments during decoding (essentially keeping the sums on the alignments and the max on the state space). This solution has not yet been tested.

[4]See the appendix of [1] for more details.

[5]Nevertheless, it can be argued that transitions between words could have been learned using the training data.

## References

[1] S. Bengio. An asynchronous hidden markov model for audio-visual speech recognition. Technical Report IDIAP-RR 02-26, IDIAP, 2002.

[2] S. Bengio and Y. Bengio. An EM algorithm for asynchronous input/output hidden markov models. In *Proceedings of the International Conference on Neural Information Processing, ICONIP*, Hong Kong, 1996.

[3] S. Dupont and J. Luettin. Audio-visual speech modelling for continuous speech recognition. *IEEE Transactions on Multimedia*, 2:141–151, 2000.

[4] R. Durbin, S. Eddy, A. Krogh, and G. Michison. *Biological Sequence Analysis: Probabilistic Models of proteins and nucleic acids*. Cambridge University Press, 1998.

[5] S. Pigeon and L. Vandendorpe. The M2VTS multimodal face database (release 1.00). In *Proceedings of the First International Conference on Audio- and Video-based Biometric Person Authentication ABVPA*, 1997.

[6] Laurence R. Rabiner. A tutorial on hidden markov models and selected applications in speech recognition. *Proceedings of the IEEE*, 77(2):257–286, 1989.

[7] W. H. Sumby and I. Pollak. Visual contributions to speech intelligibility in noise. *Journal of the Acoustical Society of America*, 26:212–215, 1954.

[8] A. Q. Summerfield. Lipreading and audio-visual speech perception. *Philosophical Transactions of the Royal Society of London, Series B*, 335:71–78, 1992.

[9] A. Varga, H.J.M. Steeneken, M. Tomlinson, and D. Jones. The noisex-92 study on the effect of additive noise on automatic speech recognition. Technical report, DRA Speech Research Unit, 1992.
